# Learning in Markov Random Fields using Tempered Transitions

**Ruslan Salakhutdinov**
Brain and Cognitive Sciences and CSAIL
Massachusetts Institute of Technology
rsalakhu@mit.edu

## Abstract

Markov random fields (MRF's), or undirected graphical models, provide a powerful framework for modeling complex dependencies among random variables. Maximum likelihood learning in MRF's is hard due to the presence of the global normalizing constant. In this paper we consider a class of stochastic approximation algorithms of the Robbins-Monro type that use Markov chain Monte Carlo to do approximate maximum likelihood learning. We show that using MCMC operators based on tempered transitions enables the stochastic approximation algorithm to better explore highly multimodal distributions, which considerably improves parameter estimates in large, densely-connected MRF's. Our results on MNIST and NORB datasets demonstrate that we can successfully learn good generative models of high-dimensional, richly structured data that perform well on digit and object recognition tasks.

## 1 Introduction

Markov random fields (MRF's) provide a powerful tool for representing dependency structure between random variables. They have been successfully used in various application domains, including machine learning, computer vision, and statistical physics. The major limitation of MRF's is the need to compute the partition function, whose role is to normalize the joint distribution over the set of random variables. Maximum likelihood learning in MRF's is often very difficult because of the hard inference problem induced by the partition function. When modeling high-dimensional, richly structured data, the inference problem becomes much more difficult because the distribution we need to infer is likely to be highly multimodal [17]. Multimodality is common in real-world distributions, such as the distribution of natural images, in which an exponentially large number of possible image configurations have extremely low probability, but there are many very different images that occur with similar probabilities.

To date, there has been very little work addressing the problem of efficient learning in large, densely-connected MRF's that contain millions of parameters. While there exists a substantial literature on developing approximate learning algorithms for arbitrary MRF's, many of these algorithms are unlikely to work well when dealing with high-dimensional inputs. Methods that are based on replacing the likelihood term with some tractable approximations, such as pseudo-likelihood [1] or mixtures of random spanning trees [11], perform very poorly for densely-connected MRF's with strong dependency structures [3]. When using variational methods, such as loopy BP [18] and TRBP [16], learning often gets trapped in poor local optima [5, 13]. MCMC-based algorithms, including MCMC maximum likelihood estimators [3, 20] and Contrastive Divergence [4], typically suffer from high variance (or strong bias) in their estimates, and can sometimes be painfully slow. The main problem here is the inability of Markov chains to efficiently explore distributions with many isolated modes.

In this paper we concentrate on the class of stochastic approximation algorithms of the Robbins-Monro type that use MCMC to estimate the model's expected sufficient statistics, needed for maximum likelihood learning. We first show that using this class of algorithms allows us to make very rapid progress towards finding a fairly good set of parameters, even for models containing millions of parameters. Second, we show that using MCMC operators based on tempered transitions [9] enables the stochastic algorithm to better explore highly multimodal distributions, which considerably improves parameter estimates, particularly in large, densely-connected MRF's. Our results on the MNIST and NORB datasets demonstrate that the stochastic approximation algorithm together with tempered transitions can be successfully used to model high-dimensional real-world distributions.

## 2    Maximum Likelihood Learning in MRF's

Let $\mathbf{x} \in \mathcal{X}^K$ be a random vector on $K$ variables, where each $x_i$ takes on values in some discrete alphabet. Let $\phi(\mathbf{x})$ denote a $D$-dimensional vector of sufficient statistics, and let $\theta \in R^D$ be a vector of canonical parameters. The exponential family associated with sufficient statistics $\phi$ consists of the following parameterized set of probability distributions:

$$p(\mathbf{x}; \theta) = \frac{p^*(\mathbf{x})}{\mathcal{Z}(\theta)} = \frac{1}{\mathcal{Z}(\theta)} \exp\left(\theta^\top \phi(\mathbf{x})\right), \qquad \mathcal{Z}(\theta) = \sum_{\mathbf{x}} \exp\left(\theta^\top \phi(\mathbf{x})\right), \tag{1}$$

where $p^*(\cdot)$ denotes the unnormalized probability distribution and $\mathcal{Z}(\theta)$ is the partition function. For example, consider the following binary pairwise MRF. Given a graph $G = (V, E)$ with vertices $V$ and edges $E$, the probability distribution over a binary random vector $\mathbf{x} \in \{0, 1\}^K$ is given by:

$$p(\mathbf{x}; \theta) = \frac{1}{\mathcal{Z}(\theta)} \exp\left(\theta^\top \phi(\mathbf{x})\right) = \frac{1}{\mathcal{Z}(\theta)} \exp\left(\sum_{(i,j) \in E} \theta_{ij} x_i x_j + \sum_{i \in V} \theta_i x_i\right). \tag{2}$$

The derivative of the log-likelihood for an observation $\mathbf{x}_0$ with respect to parameter vector $\theta$ can be obtained from Eq. 1:

$$\frac{\partial \log p(\mathbf{x}_0; \theta)}{\partial \theta} = \phi(\mathbf{x}_0) - \mathrm{E}_{p(\mathbf{x};\theta)}[\phi(\mathbf{x})], \tag{3}$$

where $\mathrm{E}_P[\cdot]$ denotes an expectation with respect to distribution $P$. Except for simple models such as the tree structured graphs exact maximum likelihood learning is intractable, because exact computation of the expectation $\mathrm{E}_{p(\mathbf{x};\theta)}[\cdot]$ takes time that is exponential in the treewidth of the graph[1].

One approach is to learn model parameters by maximizing the pseudo-likelihood (PL) [1], which replaces the likelihood with a tractable product of conditional probabilities:

$$P_{\mathrm{PL}}(\mathbf{x}_0; \theta) = \prod_{k=1}^{K} p(x_k | \mathbf{x}_{0,-k}; \theta), \tag{4}$$

where $\mathbf{x}_{0,-k}$ denotes an observation vector $\mathbf{x}_0$ with $x_k$ omitted. Pseudo-likelihood provides good estimates for weak dependence, when $p(x_k | \mathbf{x}_{-k}) \approx p(x_k)$, or when it well approximates the true likelihood function. For MRF's with strong dependence structure, it is unlikely to work well.

Another approach, called the MCMC maximum likelihood estimator (MCMC-MLE) [3], has been shown to sometimes provide considerably better results than PL [3, 20]. The key idea is to use importance sampling to approximate the model's partition function. Consider running a Markov chain to obtain samples $\mathbf{x}^{(1)}, \mathbf{x}^{(2)}, ..., \mathbf{x}^{(n)}$ from some fixed proposal distribution $p(\mathbf{x}; \psi)$[2]. These samples can be used to approximate the log-likelihood ratio for an observation $\mathbf{x}_0$:

$$L(\theta) = \log \frac{p(\mathbf{x}_0; \theta)}{p(\mathbf{x}_0; \psi)} = (\theta - \psi)^\top \phi(\mathbf{x}_0) - \log \frac{\mathcal{Z}(\theta)}{\mathcal{Z}(\psi)} \tag{5}$$

$$\approx (\theta - \psi)^\top \phi(\mathbf{x}_0) - \log \frac{1}{n} \sum_{i=1}^{n} e^{(\theta - \psi)^\top \phi(\mathbf{x}^{(i)})} = L_n(\theta), \tag{6}$$

**Algorithm 1** Stochastic Approximation Procedure.
---
1: Given an observation $\mathbf{x}_0$. Randomly initialize $\theta^1$ and $M$ sample particles $\{\mathbf{x}^{1,1}, ...., \mathbf{x}^{1,M}\}$.
2: **for** $t = 1 : T$ (number of iterations) **do**
3:    **for** $m = 1 : M$ (number of parallel Markov chains) **do**
4:       Sample $\mathbf{x}^{t+1,m}$ given $\mathbf{x}^{t,m}$ using transition operator $T_{\theta^t}(\mathbf{x}^{t+1,m} \leftarrow \mathbf{x}^{t,m})$.
5:    **end for**
6:    Update: $\theta^{t+1} = \theta^t + \alpha_t \left[ \phi(\mathbf{x}_0) - \frac{1}{M} \sum_{m=1}^{M} \phi(\mathbf{x}^{t+1,m}) \right]$.
7:    Decrease $\alpha_t$.
8: **end for**
---

where we used the approximation: $\frac{\mathcal{Z}(\theta)}{\mathcal{Z}(\psi)} = \sum_{\mathbf{x}} e^{(\theta-\psi)^\top \phi(\mathbf{x})} p(\mathbf{x}; \psi) \approx \frac{1}{n} \sum_{i=1}^{n} e^{(\theta-\psi)^\top \phi(\mathbf{x}^{(i)})}$. Provided our Markov chain is ergodic, it can be shown that $L_n(\theta) \to L(\theta)$ for all $\theta$. It can further be shown that, under the "usual" regularity conditions, if $\hat{\theta}_n$ maximizes $L_n(\theta)$ and $\theta^*$ maximizes $L(\theta)$, then $\hat{\theta}_n \xrightarrow{a.s.} \theta^*$. This implies that as the number of samples $n$, drawn from our proposal distributions, goes to infinity, MCMC-MLE will converge to the true maximum likelihood estimator. While this estimator provides nice asymptotic convergence guarantees, it performs very poorly in practice, particularly when the parameter vector $\theta$ is high-dimensional. In high-dimensional spaces, the variance of an estimator $L_n(\theta)$ will be very large, or possibly infinite, unless the proposal distribution $p(\mathbf{x}; \psi)$ is a near-perfect approximation to $p(\mathbf{x}; \theta)$. While there have been some attempts to improve MCMC-MLE by considering a mixture of proposal distributions [20], they do not fix the problem when learning MRF's with millions of parameters.

## 3 Stochastic Approximation Procedure (SAP)

We now consider a stochastic approximation procedure that uses MCMC to estimate the model's expected sufficient statistics. SAP belongs to the general class of well-studied stochastic approximation algorithms of the Robbins-Monro type [19, 12]. The algorithm itself dates back to 1988 [19], but only recently it has been shown to work surprisingly well when training large MRF's, including restricted Boltzmann machines [15] and deep Boltzmann machines [14, 13].

The idea behind learning a parameter vector $\theta$ using SAP is straightforward. Let $\mathbf{x}_0$ be our observation. Then the state and the parameters are updated sequentially:

$$\theta^{t+1} = \theta^t + \alpha_t \left[ \phi(\mathbf{x}_0) - \phi(\mathbf{x}^{t+1}) \right], \quad \text{where} \quad \mathbf{x}^{t+1} \sim T_{\theta^t}(\mathbf{x}^{t+1} \leftarrow \mathbf{x}^t). \tag{7}$$

Given $\mathbf{x}^t$, we sample a new state $\mathbf{x}^{t+1}$ using the transition operator $T_{\theta^t}(\mathbf{x}^{t+1} \leftarrow \mathbf{x}^t)$ that leaves $p(\cdot; \theta^t)$ invariant. A new parameter $\theta^{t+1}$ is then obtained by replacing the intractable expectation $\mathrm{E}_{p(\mathbf{x}; \theta^t)}[\phi(\mathbf{x})]$ with $\phi(\mathbf{x}^{t+1})$. In practice, we typically maintain a set of $M$ sample points $X^t = \{\mathbf{x}^{t,1}, ...., \mathbf{x}^{t,M}\}$, which we will often refer to as sample particles. In this case, the intractable model's expectation is replaced by the sample average $1/M \sum_{m=1}^{M} \phi(\mathbf{x}^{t+1,m})$. The procedure is summarized in Algorithm 1.

One important property of this algorithm is that just like MCMC-MLE, it can be shown to asymptotically converge to the maximum likelihood estimator $\theta^*$.[3] In particular, for fully visible discrete MRF's, if one uses a Gibbs transition operator and the learning rate is set to $\alpha_t = \frac{1}{(t+1)U}$, where $U$ is a positive constant, such that $U > 2KC_0C_1$, then $\theta^t \xrightarrow{a.s.} \theta^*$ (see Theorem 4.1 of [19]). Here $K$ is the dimensionality of $\mathbf{x}$, $C_0 = \max\{||\phi(\mathbf{x}_0) - \phi(\mathbf{x})||; \mathbf{x} \in \mathcal{X}^K\}$ is the largest magnitude of the gradient, and $C_1$ is the maximum variation of $\phi$ when one changes the values of a single component only: $C_1 = \max\{||\phi(\mathbf{x}) - \phi(\mathbf{y})||; \mathbf{x}, \mathbf{y} \in \mathcal{X}^K, k \in \{1, ..., K\}, \mathbf{y}_{-k} = \mathbf{x}_{-k}\}$.

The proof of convergence relies on the following simple decomposition. First, let $S(\theta)$ denote the true gradient of the log-likelihood function: $S(\theta) = \frac{\partial \log p(\mathbf{x}_0; \theta)}{\partial \theta} = \phi(\mathbf{x}_0) - \mathrm{E}_{p(\mathbf{x}; \theta)}[\phi(\mathbf{x})]$. The parameter update rule then takes the following form:

$$\begin{aligned} \theta^{t+1} = \theta^t + \alpha_t \left[ \phi(\mathbf{x}_0) - \phi(\mathbf{x}^{t+1}) \right] &= \theta^t + \alpha_t S(\theta^t) + \alpha_t \left[ \mathrm{E}_{p(\mathbf{x}; \theta)}[\phi(\mathbf{x})] - \phi(\mathbf{x}^{t+1}) \right] \\ &= \theta^t + \alpha_t S(\theta^t) + \alpha_t \epsilon_t. \end{aligned} \tag{8}$$

**Algorithm 2** Tempered Transitions Run.

---

1: Initialize $\beta_0 < \beta_1 < ... < \beta_S = 1$. Given a current state $\mathbf{x}^S$.
2: **for** $s = S - 1 : 0$ (Forward pass) **do**
3:      Sample $\mathbf{x}^s$ given $\mathbf{x}^{s+1}$ using $T_s(\mathbf{x}^s \leftarrow \mathbf{x}^{s+1})$.
4: **end for**
5: Set $\tilde{\mathbf{x}}^0 = \mathbf{x}^0$.
6: **for** $s = 0 : S - 1$ (Backward pass) **do**
7:      Sample $\tilde{\mathbf{x}}^{s+1}$ given $\tilde{\mathbf{x}}^s$ using $\widetilde{T}_s(\tilde{\mathbf{x}}^{s+1} \leftarrow \tilde{\mathbf{x}}^s)$.
8: **end for**
9: Accept a new state $\tilde{\mathbf{x}}^S$ with probability: $\min \left[ 1, \prod_{s=1}^{S} p^*(\mathbf{x}_s)^{\beta_{s-1} - \beta_s} \, p^*(\tilde{\mathbf{x}}_s)^{\beta_s - \beta_{s-1}} \right]$.

---

The first term (rhs. of Eq. 8) is the discretization of the ordinary differential equation $\dot{\theta} = S(\theta)$. The algorithm is therefore a perturbation of this discretization with the noise term $\epsilon_t$. The proof proceeds by showing that the noise term is not too large. Intuitively, as the learning rate becomes sufficiently small compared to the mixing rate of the Markov chain, the chain will stay close to the stationary distribution, even if it is only run for a few MCMC steps per parameter update. This, in turn, will ensure that the noise term $\epsilon_t$ goes to zero.

When looking at the behavior of this algorithm in practice, we find that initially it makes very rapid progress towards finding a sensible region in the parameter space. However, as the algorithm begins to capture the multimodality of the data distribution, the Markov chain tends to mix poorly, producing highly correlated samples for successive parameter updates. This often leads to poor parameter estimates, especially when modeling complex, high-dimensional distributions. The main problem here is the inability of the Markov chain to efficiently explore a distribution with many isolated modes. However, the transition operators $T_{\theta^t}(\mathbf{x}^{t+1} \leftarrow \mathbf{x}^t)$ used in the stochastic approximation algorithm do not necessarily need to be simple Gibbs or Metropolis-Hastings updates to guarantee almost sure convergence. Instead, we propose to use MCMC operators based on tempered transitions [9] that can more efficiently explore highly multimodal distributions. In addition, implementing tempered transitions requires very little extra work beyond the implementation of the Gibbs sampler.

## 3.1 Tempered Transitions

Suppose that our goal is to sample from $p(\mathbf{x}; \theta)$. We first define a sequence of intermediate probability distributions: $p_0, ..., p_S$, with $p_S = p(\mathbf{x}; \theta)$ and $p_0$ being more spread out and easier to sample from than $p_S$. Constructing a suitable sequence of intermediate probability distributions will in general depend on the problem. One general way to define this sequence is:

$$p_s(\mathbf{x}) \quad \propto \quad p^*(\mathbf{x}; \theta)^{\beta_s}, \tag{9}$$

with "inverse temperatures" $\beta_0 < \beta_1 < ... < \beta_S = 1$ chosen by the user. For each $s = 1, .., S-1$ we define a transition operator $T_s(\mathbf{x}' \leftarrow \mathbf{x})$ that leaves $p_s$ invariant. In our implementation $T_s(\mathbf{x}' \leftarrow \mathbf{x})$ is the Gibbs sampling operator. We also need to define a reverse transition operator $\widetilde{T}_s(\mathbf{x} \leftarrow \mathbf{x}')$ that satisfies the following reversibility condition for all $\mathbf{x}$ and $\mathbf{x}'$:

$$p_s(\mathbf{x}) T_s(\mathbf{x}' \leftarrow \mathbf{x}) = \widetilde{T}_s(\mathbf{x} \leftarrow \mathbf{x}') p_s(\mathbf{x}'). \tag{10}$$

If $T_s$ is reversible, then $\widetilde{T}_s$ is the same as $T_s$. Many commonly used transition operators, such as Metropolis–Hastings, are reversible. Non-reversible operators are usually composed of several reversible sub-transitions applied in sequence $T_s = Q_1...Q_K$, such as the single component updates in a Gibbs sampler. The reverse operator can be simply constructed from the same sub-transitions, but applied in the reverse order $\widetilde{T}_s = Q_K...Q_1$.

Given the current state $\mathbf{x}$ of the Markov chain, tempered transitions apply a sequence of transition operators $T_{S-1} \ldots T_0 \widetilde{T}_0 \ldots \widetilde{T}_{S-1}$ that systematically "move" the sample particle $\mathbf{x}$ from the original complex distribution to the easily sampled distribution, and then back to the original distribution. A new candidate state $\tilde{\mathbf{x}}$ is accepted or rejected based on ratios of probabilities of intermediate states. Since $p_0$ is less concentrated than $p_S$, the sample particle will have a chance to move around the state space more easily, and we may hope that the probability distribution of the resulting candidate

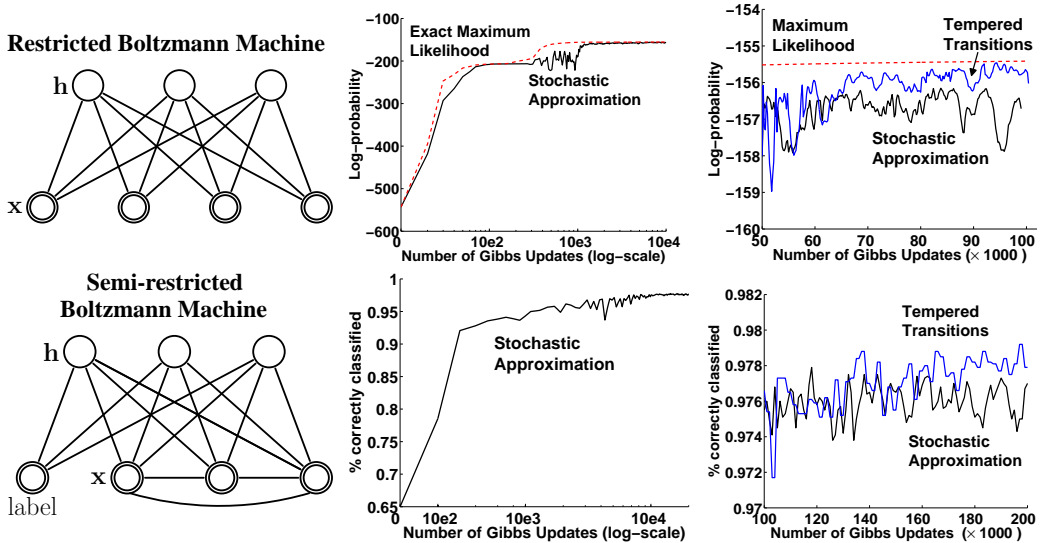

Figure 1: Experimental results on MNIST dataset. **Top:** Toy RBM with 10 hidden units. The x-axis show the number of Gibbs updates and the y-axis displays the training log-probability in nats. **Bottom:** Classification performance of the semi-restricted Boltzmann machines with 500 hidden units on the full MNIST datasets.

state will be much broader than the mode in which the current start state resides. The procedure is shown in Algorithm 2. Note that there is no need to compute the normalizing constants of any intermediate distributions.

Tempered transitions can make major changes to the current state, which allows the Markov chain to produce less correlated samples between successive parameter updates. This can greatly improve the accuracy of the estimator, but is also more computationally expensive. We therefore propose to alternate between applying a more expensive tempered transitions operator and the standard Gibbs updates. We call this algorithm Trans-SAP.

## 4 Experimental Results

In our experiments we used the MNIST and NORB datasets. To speed-up learning, we subdivided datasets into minibatches, each containing 100 training cases, and updated the parameters after each minibatch. The number of sample particles used for estimating the model's expected sufficient statistics was also set to 100. For the stochastic approximation algorithm, we always apply a single Gibbs update to the sample particles. In all experiments, the learning rates were set by quickly running a few preliminary experiments and picking the learning rates that worked best on the validation set. We also use natural logarithms, providing values in nats.

### 4.1 MNIST

The MNIST digit dataset contains 60,000 training and 10,000 test images of ten handwritten digits (0 to 9), with $28 \times 28$ pixels. The dataset was binarized: each pixel value was stochastically set to 1 with probability proportional to its pixel intensity. From the training data, a random sample of 10,000 images was set aside for validation.

In our first experiment we trained a small restricted Boltzmann machine (RBM). An RBM is a particular type of Markov random field that has a two-layer architecture, in which the visible binary stochastic units $\mathbf{x}$ are connected to hidden binary stochastic units $\mathbf{h}$, as shown in Fig. 1. The probability that the model assigns to a visible vector $\mathbf{x}$ is:

$$P(\mathbf{x}; \theta) = \frac{1}{\mathcal{Z}(\theta)} \sum_{\mathbf{h}} \exp \left( \sum_{i,j} \theta_{ij} x_i h_j + \sum_i \theta_i x_i + \sum_j \theta_j h_j \right). \tag{11}$$

| Samples before<br>Tempered Transitions | Samples after<br>Tempered Transitions | Model Samples |
|:---:|:---:|:---:|

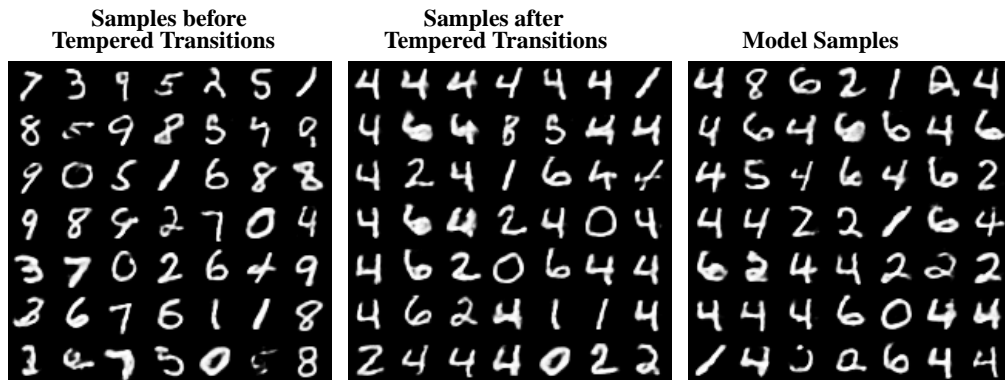

Figure 2: **Left:** Sample particles produced by the stochastic approximation algorithm after 100,000 parameter updates. **Middle:** Sample particles after applying a tempered transitions run. **Right:** Samples generated from the current model by randomly initializing all binary states and running the Gibbs sampler for 500,000 steps. After applying tempered transitions, sample particles look more like the samples generated from the current model. The images shown are the *probabilities* of the visible units given the binary states of the hidden units.

The model had 10 hidden units. This allowed us to calculate the exact value of the partition function simply by summing out the 784 visible units for each configuration of the hiddens. For the stochastic approximation procedure, the total number of parameter updates was 100,000, so the learning took about 25.6 minutes on a Pentium 4 3.00GHz machine. The learning rate was kept fixed at 0.01 for the first 10,000 parameter updates, and was then annealed as $10/(1000+t)$. For comparison, we also trained the same model using exact maximum likelihood with exactly the same learning schedule.

Perhaps surprisingly, SAP makes very rapid progress towards the maximum likelihood solution, even though the model contains 8634 free parameters. The top panel of Fig. 1 further shows that combining regular Gibbs updates with tempered transitions provides a more accurate estimator. We applied tempered transitions only during the last 50,000 Gibbs steps, alternating between 200 Gibbs updates and a single tempered transitions run that used 50 $\beta$'s spaced uniformly from 1 to 0.9. The acceptance rate for the tempered transitions was about 0.8. To be fair, we compared different algorithms based on the total number of Gibbs steps. For SAP, parameters were updated after each Gibbs step (see Algorithm 1), whereas for Trans-SAP, parameters were updated after each Gibbs update but not during the tempered transitions run[4]. Hence Trans-SAP took slightly less computer time compared to the plain SAP. Pseudo-likelihood and MCMC maximum likelihood estimators perform quite poorly, even for this small toy problem.

In our second experiment, we trained a larger semi-restricted Boltzmann machine that contained 705,622 parameters. In contrast to RBM's, the visible units in this model form a fully connected pairwise binary MRF (see Fig. 1, bottom left panel). The model had 500 hidden units and was trained to model the joint probability distribution over the digit images and labels. The total number of Gibbs updates was set to 200,000, so the learning took about 19.5 hours. The learning rate was kept fixed at 0.05 for the first 50,000 parameter updates, and was then decreased as $100/(2000+t)$.

The bottom panel of Fig. 1 shows classification performance on the full MNIST test set. As expected, SAP makes very rapid progress towards finding a good setting of the parameter values. Using tempered transitions further improves classification performance. As in our previous experiment, tempered transitions were only applied during the last 100,000 Gibbs updates, alternating between 1000 Gibbs updates and a single tempered transitions run that used 500 $\beta$'s spaced uniformly from 1 to 0.9. The acceptance rate was about 0.7. After learning was complete, in addition to classification performance, we also estimated the log-probability that both models assigned to the test data. To estimate the models' partition functions, we used Annealed Importance Sampling [10, 13] – a technique that is very similar to tempered transitions. The plain stochastic approximation algorithm achieved an average test log-probability of -87.12 per image, whereas Trans-SAP achieved a considerably better average test log-probability of -85.91.

| Training Samples | Model trained with Tempered Transitions | Model trained without Tempered Transitions |
|---|---|---|

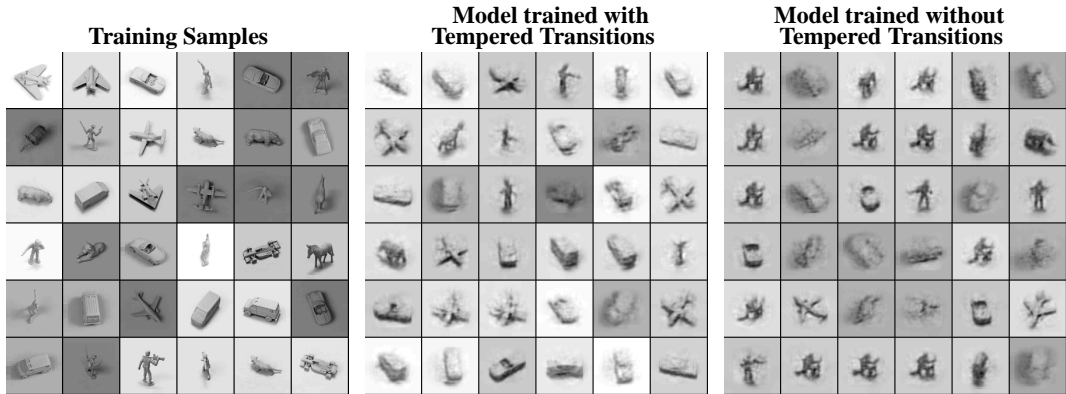

Figure 3: Results on the NORB dataset. **Left:** Random samples from the training set. Samples generated from the two RBM models, trained using SAP with (**Middle**) and without (**Right**) tempered transitions. Samples were generated by running the Gibbs sampler for 100,000 steps.

To get an intuitive picture of how tempered transitions operate, we looked at the sample particles before and after applying a tempered transitions run. Figure 2 shows sample particles after 100,000 parameter updates. Observe that the particles look like the real handwritten digits. However, a run of tempered transitions reveals that the current model is very unbalanced, with more probability mass placed on images of four. To further test whether the "refreshed" particles were representative of the current model, we generated samples from the current model by randomly initializing binary states of the visible and hidden units, and running the Gibbs sampler for 500,000 steps. Clearly, the refreshed particles look more like the samples generated from the true model. This in turn allows Trans-SAP to better estimate the model's expected sufficient statistics, which greatly facilitates learning a better generative model.

## 4.2   NORB

Results on MNIST show that the stochastic approximation algorithm works well on the relatively simple task of handwritten digit recognition. In this section we present results on a considerably more difficult dataset. NORB [6] contains images of 50 different 3D toy objects with 10 objects in each of five generic classes: planes, cars, trucks, animals, and humans. The training set contains 24,300 stereo image pairs of 25 objects, whereas the test set contains 24,300 stereo pairs of the remaining, different 25 objects. The goal is to classify each object into its generic class. From the training data, 4,300 cases were set aside for validation.

Each image has 96×96 pixels with integer greyscale values in the range [0,255]. We further reduced the dimensionality of each image from 9216 down to 4488 by using larger pixels around the edges of the image[5]. We also augmented the training data with additional *unlabeled* data by applying simple pixel translations, creating a total of 1,166,400 training instances. To deal with raw pixel data, we followed the approach of [8] by first learning a Gaussian-binary RBM with 4000 hidden units, and then treating the the activities of its hidden layer as "preprocessed" data. The model was trained using contrastive divergence learning for 500 epochs. The learned low-level RBM effectively acts as a preprocessor that transforms greyscale images into 4000-dimensional binary vectors, which we use as the input for training our models.

We proceeded to training an RBM with 4000 hidden units using binary representations learned by the preprocessor module[6]. The RBM, containing over 16 million parameters, was trained in a completely unsupervised way. The total number of Gibbs updates was set to 400,000. The learning rate was kept fixed at 0.01 for the first 100,000 parameter updates, and was then annealed as $100/(1000 + t)$. Similar to the previous experiments, tempered transitions were applied during the last 200,000 Gibbs updates, alternating between 1000 Gibbs updates and a single tempered transitions run that used 1000 $\beta$'s spaced uniformly from 1 to 0.9.

Figure 3 shows samples generated from two models, trained using stochastic approximation with and without tempered transitions. Both models were able to learn a lot of regularities in this high-dimensional, highly-structured data, including various object classes, different viewpoints and lighting conditions. The plain stochastic approximation algorithm produced a very unbalanced model with a large fraction of the model's probability mass placed on images of humans. Using tempered transitions allowed us to learn a better and more balanced generative model, including the lighting effects. Indeed, the plain SAP achieved a test log-probability of -611.08 per image, whereas Trans-SAP achieved a test log-probability of -598.58.

We also tested the classification performance of both models simply by fitting a logistic regression model to the labeled data (using only the 24300 labeled training examples without any translations) using the top-level hidden activities as inputs. The model trained by SAP achieved an error rate of 8.7%, whereas the model trained using Trans-SAP reduced the error rate down to 8.4%. This is compared to 11.6% achieved by SVM's, 22.5% achieved by logistic regression applied directly in the pixel space, and 18.4% achieved by K-nearest neighbors [6].

## 5 Conclusions

We have presented a class of stochastic approximation algorithms of the Robbins-Monro type that can be used to efficiently learn parameters in large densely-connected MRF's. Using MCMC operators based on tempered transitions allows the stochastic approximation algorithm to better explore highly multimodal distributions, which in turn allows us to learn good generative models of hand-written digits and 3D objects in a reasonable amount of computer time.

In this paper we have concentrated only on using tempered transition operators. There exist a variety of other methods for sampling from distributions with many isolated modes, including simulated tempering [7] and parallel tempering [3], all of which can be incorporated into SAP. In particular, the concurrent work of [2] employs parallel tempering techniques to imrpove mixing in RBM's. There are, however, several advantages of using tempered transitions over other existing methods. First, tempered transitions do not require specifying any extra variables, such as the approximate values of normalizing constants of intermediate distributions, which are needed for the simulated tempering method. Second, tempered transitions have modest memory requirements, unlike, for example, parallel tempering, since the acceptance rule can be computed on the fly as the intermediate states are generated. Finally, the implementation of tempered transitions requires almost no extra work beyond implementing the Gibbs sampler, and can be easily integrated into existing code.

**Acknowledgments**

We thank Vinod Nair for sharing his code for blurring and translating NORB images. This research was supported by NSERC.

## Footnotes

[1]For many interesting models considered in this paper exact computation of $\mathrm{E}_{p(\mathbf{x};\theta)}[\cdot]$ takes time that is exponential in the dimensionality of $\mathbf{x}$.

[2]We will also assume that $p(\mathbf{x}; \psi) \neq 0$ whenever $p(\mathbf{x}; \theta) \neq 0$, $\forall \theta$.

[3]One necessary condition for almost sure convergence requires the learning rate to decrease with time, so that $\sum_{t=0}^{\infty} \alpha_t = \infty$ and $\sum_{t=0}^{\infty} \alpha_t^2 < \infty$.

[4]This reduced the total number of parameter updates from $100,000$ to $50,000 + 50,000 * 2/3 = 83,333$.

[5]The dimensionality of each training vector, representing a stereo pair, was 2×4488 = 8976.

[6]The resulting model is effectively a Deep Belief Network with two hidden layers.

## References

[1] J. Besag. Efficiency of pseudolikelihood estimation for simple Gaussian fields. *Biometrica*, 64:616–618, 1977.

[2] G. Desjardins, A. Courville, Y. Bengio, P. Vincent, and O. Delalleau. Tempered Markov chain Monte Carlo for training of restricted Boltzmann machines. Technical Report 1345, University of Montreal, 2009.

[3] C. Geyer. Markov chain Monte Carlo maximum likelihood. In *Computing Science and Statistics*, pages 156–163, 1991.

[4] G. Hinton. Training products of experts by minimizing contrastive divergence. *Neural Computation*, 14(8):1711–1800, 2002.

[5] A. Kulesza and F. Pereira. Structured learning with approximate inference. In *NIPS*, 2007.

[6] Y. LeCun, F. J. Huang, and L. Bottou. Learning methods for generic object recognition with invariance to pose and lighting. In *CVPR (2)*, pages 97–104, 2004.

[7] E. Marinari and G. Parisi. Simulated tempering: A new Monte Carlo scheme. *Europhysics Letters*, 19:451–458, 1992.

[8] V. Nair and G. Hinton. Implicit mixtures of restricted Boltzmann machines. In *Advances in Neural Information Processing Systems*, volume 21, 2009.

[9] R. Neal. Sampling from multimodal distributions using tempered transitions. *Statistics and Computing*, 6:353–366, 1996.

[10] R. Neal. Annealed importance sampling. *Statistics and Computing*, 11:125–139, 2001.

[11] P. Pletscher, C. Ong, and J. Buhmann. Spanning tree approximations for conditional random fields. In *Proceedings of the International Conference on Artificial Intelligence and Statistics*, volume 5, 2009.

[12] H. Robbins and S. Monro. A stochastic approximation method. *Ann. Math. Stat.*, 22:400–407, 1951.

[13] R. Salakhutdinov. Learning and evaluating Boltzmann machines. Technical Report UTML TR 2008-002, Department of Computer Science, University of Toronto, 2008.

[14] R. Salakhutdinov and G. Hinton. Deep Boltzmann machines. In *Proceedings of the International Conference on Artificial Intelligence and Statistics*, volume 5, pages 448–455, 2009.

[15] T. Tieleman. Training restricted Boltzmann machines using approximations to the likelihood gradient. In *Machine Learning, Proceedings of the Twenty-first International Conference (ICML 2008)*. ACM, 2008.

[16] M. Wainwright, T. Jaakkola, and A. Willsky. Tree-reweighted belief propagation algorithms and approximate ML estimation by pseudo-moment matching. In *AI and Statistics*, volume 9, 2003.

[17] M. Welling and C. Sutton. Learning in Markov random fields with Contrastive Free Energies. In *Proceedings of the International Conference on Artificial Intelligence and Statistics*, volume 10, 2005.

[18] J. S. Yedidia, W. T. Freeman, and Y. Weiss. Constructing free-energy approximations and generalized belief propagation algorithms. *IEEE Transactions on Information Theory*, 51(7):2282–2312, 2005.

[19] L. Younes. Estimation and annealing for Gibbsian fields. *Ann. Inst. Henri Poincaré (B)*, 24(2):269–294, 1988.

[20] S. Zhu and X. Liu. Learning in Gibbsian fields: How accurate and how fast can it be? In *Proceedings of the IEEE Conference on Computer Vision and Pattern Recognition (CVPR-00)*, pages 2–9. IEEE, 2000.

